# Minimum Average Cost Clustering

**Kiyohito Nagano**
Institute of Industrial Science
University of Tokyo, Japan
nagano@sat.t.u-tokyo.ac.jp

**Yoshinobu Kawahara**
The Institute of Scientific and Industrial Research
Osaka University, Japan
kawahara@ar.sanken.osaka-u.ac.jp

**Satoru Iwata**
Research Institute for Mathematical Sciences
Kyoto University, Japan
iwata@kurims.kyoto-u.ac.jp

## Abstract

A number of objective functions in clustering problems can be described with submodular functions. In this paper, we introduce the minimum average cost criterion, and show that the theory of intersecting submodular functions can be used for clustering with submodular objective functions. The proposed algorithm does not require the number of clusters in advance, and it will be determined by the property of a given set of data points. The minimum average cost clustering problem is parameterized with a real variable, and surprisingly, we show that all information about optimal clusterings for all parameters can be computed in polynomial time in total. Additionally, we evaluate the performance of the proposed algorithm through computational experiments.

## 1 Introduction

A clustering of a finite set $V$ of data points is a partition of $V$ into subsets (called clusters) such that data points in the same cluster are similar to each other. Basically, a clustering problem asks for a partition $\mathcal{P}$ of $V$ such that the intra-cluster similarity is maximized and/or the inter-cluster similarity is minimized. The clustering of data is one of the most fundamental unsupervised learning problems. We use a criterion function defined on all partitions of $V$, and the clustering problem becomes that of finding a partition of $V$ that minimizes the clustering cost under some constraints. Suppose that the inhomogeneity of subsets of the data set is measured by a nonnegative set function $f : 2^V \to \mathbb{R}$ with $f(\emptyset) = 0$, where $2^V$ denotes the set of all subsets of $V$, and the clustering cost of a partition $\mathcal{P} = \{S_1, S_2, \ldots, S_k\}$ is defined by $f[\mathcal{P}] = f(S_1) + \cdots + f(S_k)$. A number of set functions that represent the inhomogeneity, including cut functions of graphs and entropy functions, are known to be submodular [3, 4]. Throughout of this paper, we suppose that $f$ is *submodular*, that is, $f(S) + f(T) \geq f(S \cup T) + f(S \cap T)$ for all $S, T \subseteq V$. A submodular function is known to be a discrete counterpart of a convex function, and in recent years, its importance has been recognized in the field of machine learning.

For any given integer $k$ with $1 \leq k \leq n$, where $n$ is the number of points in $V$, a partition $\mathcal{P}$ of $V$ is called a $k$-partition if there are exactly $k$ nonempty elements in $\mathcal{P}$, and is called an optimal $k$-clustering if $\mathcal{P}$ is a $k$-partition that minimizes the cost $f[\mathcal{P}]$ among all $k$-partitions. A problem of finding an optimal $k$-clustering is widely studied in combinatorial optimization and various fields, and it is recognized as a natural formulation of a clustering problem [8, 9, 10]. Even if $f$ is a cut function of a graph, which is submodular and *symmetric*, that is, $f(V - S) = f(S)$ for all $S \subseteq V$, this problem is known to be NP-hard unless $k$ can be regarded as a constant [5]. Zhao *et al.* [17] and Narasimhan *et al.* [10] dealt with the case when $f$ is submodular and symmetric. Zhao *et al.* [17] gave a $2(1 - 1/k)$-approximation algorithm using Queyranne's symmetric submodular function minimization algorithm [13]. Narasimhan *et al.* [10] showed that Queyranne's algorithm can be used

for clustering problems with some specific natural criteria. For a general submodular function and a small constant $k$, constant factor approximation algorithms for optimal $k$-clusterings are designed in [12, 18]. In addition, balanced clustering problems with submodular costs are studied in [8, 9].

Generally speaking, it is difficult to find an optimal $k$-clustering for any given $k$ because the optimization problem is NP-hard even for simple special cases. Furthermore, the number of clusters has to be determined in advance, regardless of the property of the data points, or an additional computation is required to find a proper number of clusters via some method like cross-validation. In this paper, we introduce a new clustering criterion to resolve the above shortcomings of previous approaches [10]. In the minimum average cost (MAC) clustering problem we consider, the objective function is the average cost of a partition $\mathcal{P}$ which combines the clustering cost $f[\mathcal{P}]$ and the number of clusters $|\mathcal{P}|$. Now the number of clusters is not pre-determined, but it will be determined automatically by solving the combinatorial optimization problem. We argue that the MAC clustering problem represents a natural clustering criterion. In this paper, we show that the Dilworth truncation of an intersecting submodular function [2] (see also Chapter II of Fujishige [4] and Chapter 48 of Schrijver [14]) can be used to solve the clustering problem *exactly* and *efficiently*. To the best of our knowledge, this is the first time that the theory of intersecting submodular functions is used for clustering. The MAC clustering problem can be parameterized with a real-valued parameter $\beta \geq 0$, and the problem with respect to $\beta$ asks for a partition $\mathcal{P}$ of $V$ that minimizes the average cost under a constraint $|\mathcal{P}| > \beta$. The main contribution of this paper is a polynomial time algorithm that solves the MAC clustering problem exactly for any given parameter $\beta$. This result is in stark contrast to the NP-hardness of the optimal $k$-clustering problems. Even more surprisingly, our algorithm computes all information about MAC clusterings for all parameters in polynomial time in total.

In the case where $f$ is a cut function of a graph, there are some related works. If $f$ is a cut function and $\beta = 1$, the optimal value of the MAC clustering problem coincides with the strength of a graph [1]. In addition, the computation of the principal sequence of partitions of a graph [7] is a special case of the parametrized MAC clustering problem in an implicit way.

This paper is organized as follows. In Section 2, we formulate the minimum average cost clustering problem, and show a structure property of minimum average cost clusterings. In Section 3, we propose a framework of our algorithm for the minimum average cost clustering problem. In Section 4, we explain the basic results on the theory of intersecting submodular functions, and describe the Dilworth truncation algorithm which is used in Section 3 as a subroutine. Finally, we show the result of computational experiments in Section 5, and give concluding remarks in Section 6.

## 2 Minimum Average Cost Clustering

In this section, we give a definition of minimum average cost clusterings. After that, we show a structure property of them. Let $V$ be a given set of $n$ data points, and let $f : 2^V \rightarrow \mathbb{R}$ be a nonnegative submodular function with $f(\emptyset) = 0$, which is not necessarily symmetric. For each subset $S \subseteq V$, the value $f(S)$ represents the inhomogeneity of data points in $S$. For a partition $\mathcal{P} = \{S_1, \ldots, S_k\}$, the clustering cost is defined by $f[\mathcal{P}] = f(S_1) + \cdots + f(S_k)$. We will introduce the minimum average cost criterion in order to make consideration of both the clustering cost $f[\mathcal{P}]$ and the number of clusters $|\mathcal{P}|$.

### 2.1 Definition

Consider a $k$-partition $\mathcal{P}$ of $V$ with $k > 1$, and compare $\mathcal{P}$ with a trivial partition $\{V\}$ of $V$. Then, the number of clusters has increased by $k - 1$ and the clustering cost has increased by $f[\mathcal{P}] + c$, where $c$ is a constant. Therefore, it is natural to define an average cost of $\mathcal{P}$ by $f[\mathcal{P}]/(|\mathcal{P}| - 1)$. Suppose that $\mathcal{P}^*$ is a partition of $V$ that minimizes the average cost among all partitions $\mathcal{P}$ of $V$ with $|\mathcal{P}| > 1$. Remark that the number of clusters of $\mathcal{P}^*$ is determined not by us, but by the property of the given data set. Therefore, it may be said that $\mathcal{P}^*$ is a natural clustering.

More generally, using a parameter $\beta \in [0, n) = \{\tau \in \mathbb{R} : 0 \leq \tau < n\}$, we define an extended average cost by $f[\mathcal{P}]/(|\mathcal{P}| - \beta)$. For any parameter $\beta \in [0, n)$, we consider the minimum average cost (MAC) clustering problem

$$\lambda(\beta) := \min_{\mathcal{P}} \{f[\mathcal{P}]/(|\mathcal{P}| - \beta) : \mathcal{P} \text{ is a partition of } V, |\mathcal{P}| > \beta\}. \tag{1}$$

Let us say that a partition $\mathcal{P}$ is a $\beta$-MAC clustering if $\mathcal{P}$ is optimal for the problem (1) with respect to $\beta \in [0, n)$. Naturally, the case where $\beta = 1$ is fundamental. Furthermore, we can expect finer clusterings for relatively large parameters. The problem (1) and the optimal $k$-clustering problem [10] are closely related.

**Proposition 1.** *Let $\mathcal{P}$ be a $\beta$-MAC clustering for some $\beta \in [0, n)$, and set $k := |\mathcal{P}|$. Then we have $f[\mathcal{P}] \le f[\mathcal{Q}]$ for any $k$-partition $\mathcal{Q}$ of $V$. In other words, $\mathcal{P}$ is an optimal $k$-clustering.*

*Proof.* By definition, we have $k > \beta$ and $f[\mathcal{P}]/(k - \beta) \le f[\mathcal{Q}]/(k - \beta)$ for any $k$-partition $\mathcal{Q}$. $\quad\square$

We will show that all information about $\beta$-MAC clusterings for all parameters $\beta$ can be computed in polynomial time in total. Our algorithm requires the help of the theory of intersecting submodular functions [4, 14]. Proposition 1 says that if there exists a $\beta$-MAC clustering $\mathcal{P}$ satisfying $|\mathcal{P}| = k$, then we obtain an optimal $k$-clustering. Note that this fact is consistent with the NP-hardness of the optimal $k$-clustering problem because the information about MAC clusterings just gives a portion of the information about optimal $k$-clusterings ($k = 1, \ldots, n$).

## 2.2 Structure property

We will investigate the structure of all $\beta$-MAC clusterings. Denote by $\mathbb{R}_+$ the set of nonnegative real values. Let us choose a parameter $\beta \in [0, n)$. If $\mathcal{P}$ is a partition of $V$ satisfying $|\mathcal{P}| \le \beta$, we have $-\beta\lambda \le -|\mathcal{P}|\lambda \le f[\mathcal{P}] - |\mathcal{P}|\lambda$ for all $\lambda \in \mathbb{R}_+$. Hence the minimum average cost $\lambda(\beta)$ defined in (1) is represented as

$$
\begin{aligned}
\lambda(\beta) &= \max\{\lambda \in \mathbb{R}_+ : \lambda \le f[\mathcal{P}]/(|\mathcal{P}| - \beta) \text{ for all partition } \mathcal{P} \text{ of } V \text{ with } |\mathcal{P}| > \beta\} \\
&= \max\{\lambda \in \mathbb{R}_+ : -\beta\lambda \le f[\mathcal{P}] - |\mathcal{P}|\lambda \text{ for all partition } \mathcal{P} \text{ of } V\} \\
&= \max\{\lambda \in \mathbb{R}_+ : -\beta\lambda \le h(\lambda)\},
\end{aligned}
\tag{2}
$$

where $h : \mathbb{R}_+ \to \mathbb{R}$ is defined by

$$
h(\lambda) = \min_{\mathcal{P}}\{f[\mathcal{P}] - |\mathcal{P}|\lambda : \ \mathcal{P} \text{ is a partition of } V\} \quad (\lambda \ge 0).
\tag{3}
$$

The function $h$ does not depend on the parameter $\beta$. For $\lambda \ge 0$, we say that a partition $\mathcal{P}$ *determines $h$ at $\lambda$* if $f[\mathcal{P}] - |\mathcal{P}|\lambda = h(\lambda)$. Apparently, the minimization problem (3) is difficult to solve for any given $\lambda \ge 0$. This point will be discussed in Section 4 in detail.

Let us examine properties of the function $h$. For each partition $\mathcal{P}$ of $V$, define a linear function $h_{\mathcal{P}} : \mathbb{R}_+ \to \mathbb{R}$ as $h_{\mathcal{P}}(\lambda) = f[\mathcal{P}] - |\mathcal{P}|\lambda$. Since $h$ is the minimum of these linear functions, $h$ is a piecewise-linear concave function on $\mathbb{R}_+$. The function $h$ is illustrated in Figure 1 by the thick curve. We have $h(0) = f(V)$ because $f[\{V\}] \le f[\mathcal{P}]$ for any partition $\mathcal{P}$ of $V$. Moreover, it is easy to see that the set of singletons $\{\{1\}, \{2\}, \ldots, \{n\}\}$ determines $h$ at a sufficiently large $\lambda$. In view of (2), the minimum average cost $\lambda(\beta)$ can be obtained by solving the equation $-\beta\lambda = h(\lambda)$ (see also Figure 1). In addition, a $\beta$-MAC clustering can be characterized as follows.

**Lemma 2.** *Given a parameter $\beta \in [0, n)$, let $\mathcal{P}$ be a partition of $V$ such that $|\mathcal{P}| > \beta$ and $h(\lambda(\beta)) = f[\mathcal{P}] - |\mathcal{P}|\lambda(\beta)$. Then $\mathcal{P}$ is a $\beta$-MAC clustering.*

*Proof.* Since $-\beta\lambda(\beta) = h(\lambda(\beta)) = f[\mathcal{P}] - |\mathcal{P}|\lambda(\beta)$, we have $\lambda(\beta) = f[\mathcal{P}]/(|\mathcal{P}| - \beta)$. For any partition $\mathcal{Q}$ of $V$ with $|\mathcal{Q}| > \beta$, we have $-\beta\lambda(\beta) \le f[\mathcal{Q}] - |\mathcal{Q}|\lambda(\beta)$, and thus $\lambda(\beta) \le f[\mathcal{Q}]/(|\mathcal{Q}| - \beta)$. Therefore, $\mathcal{P}$ is a $\beta$-MAC clustering. $\quad\square$

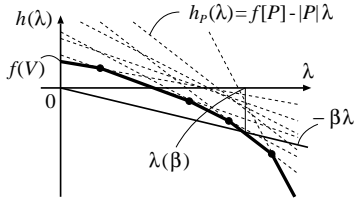

Figure 1: The function $h$

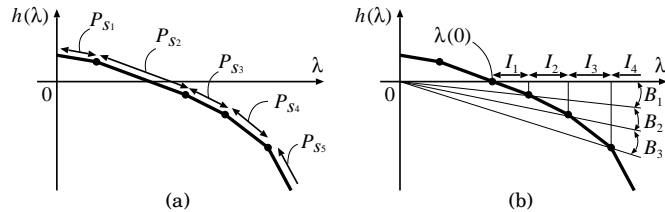

Figure 2: The structure of $h$

Now, we will present a structure property of the MAC problem (1). Suppose that the slopes of $h$ take values $-s_1 > -s_2 > \cdots > -s_d$. As $\{s_1, s_2, \ldots, s_d\} \subseteq \{1, \ldots, n\}$, we have $d \le n$. The

interval $\mathbb{R}_+$ is split into $d$ subintervals $R_1 = [0, \lambda_1)$, $R_2 = [\lambda_1, \lambda_2)$, $\ldots$, $R_d = [\lambda_{d-1}, +\infty)$ such that, for each $j = 1, \ldots, d$, the function $h$ is linear and its slope is $-s_j$ on $R_j$. Let $\mathcal{P}_{s_1}$, $\mathcal{P}_{s_2}$, $\ldots$, $\mathcal{P}_{s_d}$ be partitions of $V$ such that, for each $j = 1, \ldots, d$, the partition $\mathcal{P}_{s_j}$ determines $h$ at all $\lambda \in R_j$ (see Figure 2 (a)). In particular, $s_d = n$ and the last partition $\mathcal{P}_{s_d}$ is the set of singletons $\{\{1\}, \{2\}, \ldots, \{n\}\}$. Observe that the range $I$ of the minimum average costs $\lambda(\beta)$ is $I = [\lambda(0), +\infty)$. Suppose that $j^*$ is an index such that $\lambda(0) \in R_{j^*}$. Let $d^* = d - j^* + 1$, and let $\lambda_j^* = \lambda_{j+j^*-1}$ and $s_j^* = s_{j+j^*-1}$ for each $j = 1, \ldots, d^*$. The interval $I$ is split into $d^*$ subintervals $I_1 = [\lambda(0), \lambda_1^*)$, $I_2 = [\lambda_1^*, \lambda_2^*)$, $\ldots$, $I_{d^*} = [\lambda_{d^*-1}^*, +\infty)$. Accordingly, the domain of $\beta$ is split into $d^*$ subintervals $B_1 = [0, \beta_1)$, $B_2 = [\beta_1, \beta_2)$, $\ldots$, $B_d = [\beta_{d^*-1}, n)$, where $\beta_j = -h(\lambda_j^*)/\lambda_j^*$ for each $j = 1, \ldots, d^* - 1$. Figure 2 (b) illustrates these two sets of subintervals $\{I_1, \ldots, I_{d^*}\}$ and $\{B_1, \ldots, B_{d^*}\}$. By Lemma 2, we directly obtain the structure property of the MAC problem (1):

**Lemma 3.** *Let $j \in \{1, \ldots, d^*\}$. For any $\beta \in B_j$, the partition $\mathcal{P}_{s_j^*}$ is a $\beta$-MAC clustering.*

Lemma 3 implies that if we can find the collection $\{\mathcal{P}_{s_1}, \mathcal{P}_{s_2}, \ldots, \mathcal{P}_{s_d}\}$, then the MAC problem (1) will be completely solved. In the subsequent sections, we will give an algorithm that computes the collection $\{\mathcal{P}_{s_1}, \mathcal{P}_{s_2}, \ldots, \mathcal{P}_{s_d}\}$ in polynomial time in total.

## 3 The clustering algorithm

In this section, we present a framework of a polynomial time algorithm that finds the collection $\{\mathcal{P}_{s_1}, \mathcal{P}_{s_2}, \ldots, \mathcal{P}_{s_d}\}$ defined in §2.2. That is, our algorithm computes all the breakpoints of the piecewise linear concave function $h$ defined in (3). By Lemma 3, we can immediately construct a polynomial time algorithm that solves the MAC problem (1) completely.

The proposed algorithm uses the following procedure FINDPARTITION, which will be described in Section 4 precisely.

> **Procedure** FINDPARTITION($\lambda$): For any given $\lambda \geq 0$, this procedure computes the value $h(\lambda)$ and finds a partition $\mathcal{P}$ of $V$ that determines $h$ at $\lambda$.

We will use SFM($n$) to denote the time required to minimize a general submodular function defined on $2^V$, where $n = |V|$. Submodular function minimization can be solved in polynomial time (see [6]). Although the minimization problem (3) is apparently hard, we show that the procedure FINDPARTITION can be designed to run in polynomial time.

**Lemma 4.** *For any $\lambda \geq 0$, the procedure FINDPARTITION($\lambda$) runs in $\mathrm{O}(n \cdot \mathrm{SFM}(n))$.*

The proof of Lemma 4, which will be given in §4, utilizes the Dilworth truncation of an intersecting submodular function [4, 14].

Let us call a partition $\mathcal{P}$ of $V$ *supporting* if there exists $\lambda \geq 0$ such that $h(\lambda) = h_{\mathcal{P}}(\lambda)$. By definition, each $\mathcal{P}_{s_j}$ is supporting. In addition, for any $\lambda \geq 0$, FINDPARTITION($\lambda$) returns a supporting partition of $V$. Set $\mathcal{Q}_1 := \{V\}$ and $\mathcal{Q}_n := \{\{1\}, \{2\}, \ldots, \{n\}\}$. $\mathcal{Q}_1$ is a supporting partition of $V$ because $h(0) = f[\{V\}] = h_{\mathcal{Q}_1}(0)$, and $\mathcal{Q}_n$ is also supporting because $\mathcal{Q}_n = \mathcal{P}_{s_d}$. For a supporting partition $\mathcal{P}$ of $V$, if $|\mathcal{P}| = s_j$ for some $j \in \{1, \ldots, d\}$, then we can put $\mathcal{P}_{s_j} = \mathcal{P}$. For integers $1 \leq k < \ell \leq n$, define $R(k, \ell) = \{\lambda \in \mathbb{R}_+ : -k \geq \partial_+ h(\lambda), \text{ and } \partial_- h(\lambda) \geq -\ell\}$, where $\partial_+ h$ and $\partial_- h$ are the right and left derivatives of $h$, respectively, and we set $\partial_- h(0) = 0$. Observe that $R(k, \ell)$ is an interval in $\mathbb{R}_+$. All breakpoints of $h$ are included in $R(1, n) = \mathbb{R}_+$.

Suppose that we are given two supporting partitions $\mathcal{Q}_k$ and $\mathcal{Q}_\ell$ such that $|\mathcal{Q}_k| = k$, $|\mathcal{Q}_\ell| = \ell$ and $k < \ell$. We describe the algorithm SPLIT($\mathcal{Q}_k$, $\mathcal{Q}_\ell$), which computes the information about all breakpoints of $h$ on the interval $R(k, \ell)$. This algorithm is a recursive one. First of all, the algorithm SPLIT decides whether "$k = s_j$ and $\ell = s_{j+1}$ for some $j \in \{1, \ldots, d-1\}$" or not. Besides, if the decision is negative, the algorithm finds a supporting partition $\mathcal{Q}_m$ such that $|\mathcal{Q}_m| = m$ and $k < m < \ell$. If the decision is positive, there is exactly one breakpoint on the interior of $R(k, \ell)$, which can be given by $\mathcal{Q}_k$ and $\mathcal{Q}_\ell$. Now we show how to execute these operations. For two linear functions $h_{\mathcal{Q}_k}(\lambda)$ and $h_{\mathcal{Q}_\ell}(\lambda)$, the equality $h_{\mathcal{Q}_k}(\lambda) = h_{\mathcal{Q}_\ell}(\lambda)$ holds at $\bar{\lambda} = (f[\mathcal{Q}_\ell] - f[\mathcal{Q}_k])/(\ell - k)$. Set $\bar{h} = h_{\mathcal{Q}_k}(\bar{\lambda}) = (\ell f[\mathcal{Q}_k] - k f[\mathcal{Q}_\ell])/(\ell - k)$. Clearly, we have $h(\bar{\lambda}) \leq \bar{h}$. The algorithm SPLIT performs the procedure FINDPARTITION($\bar{\lambda}$). Consider the case where $h(\bar{\lambda}) = \bar{h}$ (see Figure 3 (a)). Then algorithm gives an affirmative answer, returns $\mathcal{Q}_k$ and $\mathcal{Q}_\ell$, and stops. Next, consider the case where $h(\bar{\lambda}) < \bar{h}$ (see Figure 3 (b)). Then the algorithm gives a negative answer, and the partition

$\mathcal{P}$ returned by FINDPARTITION is supporting and satisfies $k < |\mathcal{P}| < \ell$. We set $m = |\mathcal{P}|$ and $\mathcal{Q}_m = \mathcal{P}$. Finally, the algorithm performs SPLIT$(\mathcal{Q}_k, \mathcal{Q}_m)$ and SPLIT$(\mathcal{Q}_m, \mathcal{Q}_\ell)$.

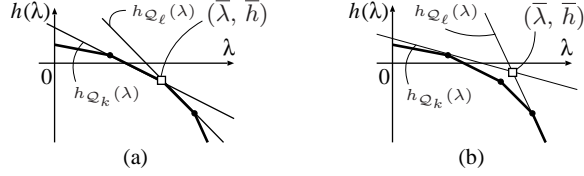

Figure 3: Two different situations in SPLIT$(\mathcal{Q}_k, \mathcal{Q}_\ell)$

The algorithm SPLIT can be summarized as follows.

---
**Algorithm** SPLIT$(\mathcal{Q}_k, \mathcal{Q}_\ell)$

   *Input :*      Supporting partitions of $V$, $\mathcal{Q}_k$ and $\mathcal{Q}_\ell$ such that $|\mathcal{Q}_k| = k$, $|\mathcal{Q}_\ell| = \ell$ and $k < \ell$.
   *Output :*   The information about all breakpoints of $h$ on the interval $R(k, \ell)$.

---
  **1**:   Set $\overline{\lambda} := (f[\mathcal{Q}_\ell] - f[\mathcal{Q}_k])/(\ell - k)$, and set $\overline{h} := (\ell f[\mathcal{Q}_k] - k f[\mathcal{Q}_\ell])/(\ell - k)$. By performing
        FINDPARTITION$(\overline{\lambda})$, compute $h(\overline{\lambda})$ and a partition $\mathcal{P}$ of $V$ that determines $h(\overline{\lambda})$.
  **2**:   If $h(\overline{\lambda}) = \overline{h}$ (positive case), return $\mathcal{Q}_k$ and $\mathcal{Q}_\ell$, and stop.
  **3**:   If $h(\overline{\lambda}) < \overline{h}$ (negative case), set $m := |\mathcal{P}|$, $\mathcal{Q}_m := \mathcal{P}$, and perform SPLIT$(\mathcal{Q}_k, \mathcal{Q}_m)$ and
        SPLIT$(\mathcal{Q}_m, \mathcal{Q}_\ell)$.

---

By performing the algorithm SPLIT$(\mathcal{Q}_1, \mathcal{Q}_n)$, where $\mathcal{Q}_1 := \{V\}$ and $\mathcal{Q}_n := \{\{1\}, \{2\}, \ldots, \{n\}\}$, the information of all breakpoints of $h$ is obtained. Therefore, the collection $\{\mathcal{P}_{s_1}, \mathcal{P}_{s_2}, \ldots, \mathcal{P}_{s_d}\}$ defined in §2.2 can be obtained. Let us show that this algorithm runs in polynomial time.

**Theorem 5.** *The collection $\{\mathcal{P}_{s_1}, \mathcal{P}_{s_2}, \ldots, \mathcal{P}_{s_d}\}$ can be computed in $\mathrm{O}(n^2 \cdot \mathsf{SFM}(n))$ time. In other words, the information of all breakpoints of $h$ can be computed in $\mathrm{O}(n^2 \cdot \mathsf{SFM}(n))$ time.*

*Proof.* By Lemma 4, it suffices to show that the number of calls of the procedure FINDPARTITION in the execution of SPLIT$(\mathcal{Q}_1, \mathcal{Q}_n)$ is $\mathrm{O}(n)$. In the algorithm, after one call of FINDPARTITION, (i) we can obtain the information about one breakpoint of $h$, or (ii) a new supporting partition $\mathcal{Q}_m$ can be obtained. Clearly, the number of breakpoints of $h$ is at most $n$. Throughout the execution of SPLIT$(\mathcal{Q}_1, \mathcal{Q}_n)$, the algorithm computes a supporting $k$-partition at most once for each $k \in \{1, \ldots, n\}$. Therefore, FINDPARTITION is called at most $2n$ times in total.    □

The main theorem of this paper directly follows from Lemma 3 and Theorem 5.

**Theorem 6.** *All information of optimal solutions to the minimum average cost clustering problem (1) for all parameters $\beta \in [0, n)$ can be computed in $\mathrm{O}(n^2 \cdot \mathsf{SFM}(n))$ time in total.*

## 4   Finding a partition

In the clustering algorithm of Section 3, we iteratively call the procedure FINDPARTITION, which computes $h(\lambda)$ defined in (3) and a partition $\mathcal{P}$ that determines $h(\lambda)$ for any given $\lambda \geq 0$. In this section, we will see that the procedure FINDPARTITION can be implemented to run in polynomial time with the aid of the Dilworth truncation of an intersecting submodular function [2], and give a proof of Lemma 4. The Dilworth truncation algorithm is sketched in the proof of Theorem 48.4 of Schrijver [14], and the algorithm described in §4.2 is based on that algorithm.

### 4.1   The Dilworth truncation of an intersecting submodular function

We start with definitions of an intersecting submodular function and the Dilworth truncation. Subsets $S, T \subseteq V$ are *intersecting* if $S \cap T \neq \emptyset$, $S \setminus T \neq \emptyset$, and $T \setminus S \neq \emptyset$. A set function $g : 2^V \to \mathbb{R}$ is *intersecting submodular* if $g(S) + g(T) \geq g(S \cup T) + g(S \cap T)$ for all intersecting subsets $S, T \subseteq V$. Clearly, the *fully* submodular function[1] $f$ is also intersecting submodular. For any $\lambda \geq 0$,

define $f_\lambda : 2^V \to \mathbb{R}$ as follows: $f_\lambda(S) = 0$ if $S = \emptyset$, and $f_\lambda(S) = f(S) - \lambda$ otherwise. It is easy to see that $f_\lambda$ is an intersecting submodular function.

For a fully submodular function $f$ with $f(\emptyset) = 0$, consider a polyhedron $\mathrm{P}(f) = \{\boldsymbol{x} \in \mathbb{R}^n : x(S) \leq f(S), \emptyset \neq \forall S \subseteq V\}$, where $x(S) = \sum_{i \in S} x_i$. The polyhedron $\mathrm{P}(f)$ is called a *submodular polyhedron*. In the same manner, for an intersecting submodular function $g$ with $g(\emptyset) = 0$, define $\mathrm{P}(g) = \{\boldsymbol{x} \in \mathbb{R}^n : x(S) \leq g(S), \emptyset \neq \forall S \subseteq V\}$. As for $\mathrm{P}(f)$, for each nonempty subset $S \subseteq V$, there exists a vector $\boldsymbol{x} \in \mathrm{P}(f)$ such that $x(S) = f(S)$ by the validity of the greedy algorithm of Edmonds [3]. On the other hand, the polyhedron $\mathrm{P}(g)$ does not necessarily satisfy such a property. Alternatively, the following property is known.

**Theorem 7** (Refer to Theorems 2.5, 2.6 of [4])**.** *Given an intersecting submodular function $g : 2^V \to \mathbb{R}$ with $g(\emptyset) = 0$, there exists a fully submodular function $\widehat{g} : 2^V \to \mathbb{R}$ such that $\widehat{g}(\emptyset) = 0$ and $\mathrm{P}(\widehat{g}) = \mathrm{P}(g)$. Furthermore, the function $\widehat{g}$ can be represented as*

$$\widehat{g}(S) = \min\{\textstyle\sum_{S \in \mathcal{P}} g(S) : \mathcal{P} \text{ is a partition of } S\}. \tag{4}$$

The function $\widehat{g}$ in Theorem 7 is called the *Dilworth truncation* of $g$. If $g$ is fully submodular, for each $S \subseteq V$, $\{S\}$ is an optimal solution to the RHS of (4) and we have $\widehat{g}(S) = g(S)$. For a general intersecting submodular function $g$, however, the computation of $\widehat{g}(S)$ is a nontrivial task.

Let us see a small example. Suppose that a fully submodular function $f : 2^{\{1, 2\}} \to \mathbb{R}$ satisfies $f(\emptyset) = 0$, $f(\{1\}) = 12$, $f(\{2\}) = 8$, and $f(\{1, 2\}) = 19$. Set $\lambda = 2$. There is no vector $x \in \mathrm{P}(f_\lambda)$ such that $x(\{1, 2\}) = f_\lambda(\{1, 2\})$. The Dilworth truncation $\widehat{f}_\lambda : 2^V \to \mathbb{R}$ defined by (4) satisfies $\widehat{f}_\lambda(S) = f_\lambda(S)$ for $S \in \{\emptyset, \{1\}, \{2\}\}$, and $\widehat{f}_\lambda(\{1, 2\}) = f_\lambda(\{1\}) + f_\lambda(\{2\}) = 16$. Observe that $\widehat{f}_\lambda$ is fully submodular and $\mathrm{P}(\widehat{f}_\lambda) = \mathrm{P}(f_\lambda)$. Figure 4 illustrates these polyhedra.

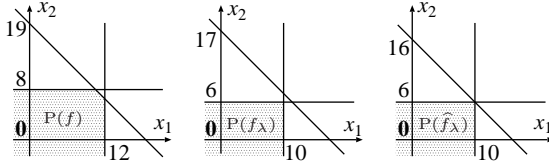

Figure 4: Polyhedra $\mathrm{P}(f)$, $\mathrm{P}(f_\lambda)$, and $\mathrm{P}(\widehat{f}_\lambda)$

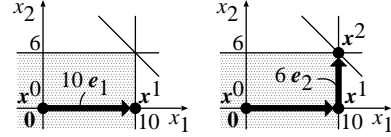

Figure 5: The greedy algorithm [3]

### 4.2 Algorithm that finds a partition

Let us fix $\lambda \geq 0$, and describe FINDPARTITION($\lambda$). In view of equations (3), (4) and the definition of $\widehat{f}_\lambda$, we obtain $h(\lambda) = \widehat{f}_\lambda(V)$ using the Dilworth truncation of $f_\lambda$. We ask for a partition $\mathcal{P}$ of $V$ satisfying $\widehat{f}_\lambda(V) = f_\lambda[\mathcal{P}]$ $(= \sum_{T \in \mathcal{P}} f_\lambda(T))$ because such a partition $\mathcal{P}$ of $V$ determines $h$ at $\lambda$.

We know that $\widehat{f}_\lambda : 2^V \to \mathbb{R}$ is submodular, but $\widehat{f}_\lambda(S) = \min\{f_\lambda[\mathcal{P}] : \mathcal{P} \text{ is a partition of } S\}$ cannot be obtained directly for each $S \subseteq V$. To evaluate $\widehat{f}_\lambda(V)$, we will use the greedy algorithm of Edmonds [3]. Denote the set of all extreme points of $\mathrm{P}(\widehat{f}_\lambda) \subseteq \mathbb{R}^n$ by $\mathrm{ex}(\mathrm{P}(\widehat{f}_\lambda))$. In the example of §4.1, we have $\mathrm{ex}(\mathrm{P}(\widehat{f}_\lambda)) = \{(10, 6)\}$. We set $\boldsymbol{x}^0 \in \mathbb{R}^n$ in such a way that $\boldsymbol{x}^0 \leq \boldsymbol{y}$ for all $\boldsymbol{y} \in \mathrm{ex}(\mathrm{P}(\widehat{f}_\lambda))$. For example, set $x_i^0 := -M$ for each $i \in V$, where $M = \lambda + \sum_{j \in V}\{|f(\{j\})| + |f(V) - f(V - \{j\})|\}$. For each $i \in V$, let $\boldsymbol{e}_i$ denote the $i$-th unit vector in $\mathbb{R}^n$.

Let $L = (i_1, \ldots, i_n)$ be any ordering of $V$, and let $V^\ell = \{i_1, \ldots, i_\ell\}$ for each $\ell = 1, \ldots, n$. Now we describe the framework of the greedy algorithm [3]. In the $\ell$-th iteration ($\ell = 1, \ldots, n$), we compute $\alpha^\ell := \max\{\alpha : \boldsymbol{x}^{\ell-1} + \alpha \cdot \boldsymbol{e}_{i_\ell} \in \mathrm{P}(\widehat{f}_\lambda)\}$ and set $\boldsymbol{x}^\ell := \boldsymbol{x}^{\ell-1} + \alpha^\ell \cdot \boldsymbol{e}_{i_\ell}$. Finally, the algorithm returns $\boldsymbol{z} := \boldsymbol{x}^n$. Figure 5 illustrates this process. By the following property, we can use the greedy algorithm to evaluate the value $h(\lambda) = \widehat{f}_\lambda(V)$.

**Theorem 8** ([3])**.** *For each $\ell = 1, \ldots, n$, we have $\widehat{f}_\lambda(V^\ell) = x^\ell(V^\ell) = z(V^\ell)$.*

Let us see that the greedy algorithm with $\widehat{f}_\lambda$ can be implemented to run in polynomial time. We discuss how to compute $\alpha^\ell$ in each iteration. Since $x^{\ell-1} \in \mathrm{P}(\widehat{f}_\lambda)$ and $\mathrm{P}(\widehat{f}_\lambda) = \mathrm{P}(f_\lambda)$, we have

$$\alpha^\ell = \max\{\alpha : \boldsymbol{x}^{\ell-1} + \alpha \cdot \boldsymbol{e}_{i_\ell} \in \mathrm{P}(f_\lambda)\} = \max\{\alpha : x^{\ell-1}(S) + \alpha \leq f_\lambda(S),\ i_\ell \in \forall S \subseteq V\}$$
$$= \min\{f(S) - x^{\ell-1}(S) - \lambda : i_\ell \in \forall S \subseteq V\}$$
$$= \min\{f(S) - x^{\ell-1}(S) - \lambda : i_\ell \in \forall S \subseteq V^\ell\}, \tag{5}$$

where the last equality holds because of the choice of the initial vector $\boldsymbol{x}^0$ (remark that $x_i^{\ell-1} = x_i^0$ for all $i \in V - V^\ell$). Hence, the value $\alpha^\ell$ can be computed by minimizing a fully submodular function. It follows from Theorem 8 that the value $h(\lambda) = \widehat{f}_\lambda(V)$ can be computed in $\mathrm{O}(n \cdot \mathsf{SFM}(n))$ time.

In addition to the value $h(\lambda)$, a partition $\mathcal{P}$ of $V$ such that $f[\mathcal{P}] - \lambda|\mathcal{P}| = h(\lambda)$ is also required. For this purpose, we modify the above greedy algorithm, and obtain the procedure FINDPARTITION.

---

**Procedure** FINDPARTITION($\lambda$)

  *Input :*    A nonnegative real value $\lambda \geq 0$.
  *Output :*  A real value $h_\lambda$ and a partition $\mathcal{P}_\lambda$ of $V$.

---

**1**:  Set $\mathcal{P}^0 := \emptyset$.
**2**:  For each $\ell = 1, \ldots, n$, do:
      Compute $\alpha^\ell = \min\{f(S) - x^{\ell-1}(S) - \lambda : i_\ell \in \forall S \subseteq V^\ell\}$;
      Find a subset $T^\ell$ such that $i_\ell \in T^\ell \subseteq V^\ell$ and $f(T^\ell) - x^{\ell-1}(T^\ell) - \lambda = \alpha^\ell$;
      Set $\boldsymbol{x}^\ell := \boldsymbol{x}^{\ell-1} + \alpha^\ell \cdot \boldsymbol{e}_{i_\ell}$, set $U^\ell := T^\ell \cup [\cup\{S : S \in \mathcal{P}^{\ell-1},\ T^\ell \cap S \neq \emptyset\}]$, and set
      $\mathcal{P}^\ell := \{U^\ell\} \cup \{S : S \in \mathcal{P}^{\ell-1},\ T^\ell \cap S = \emptyset\}$.
**3**:  Return $h_\lambda := z(V)$ and $\mathcal{P}_\lambda := \mathcal{P}^n$.

---

Basically, this procedure FINDPARTITION($\lambda$) is the same algorithm as the above greedy algorithm. But now, we compute $\mathcal{P}^\ell$ in each iteration. Figure 6 shows the computation of $\mathcal{P}^\ell$ in the $\ell$-th iteration of the procedure FINDPARTITION($\lambda$). For each $\ell = 1, \ldots, n$, $\mathcal{P}^\ell$ is a partition of $V^\ell = \{i_1, \ldots, i_\ell\}$. Thus, $\mathcal{P}_\lambda$ is a partition of $V$.

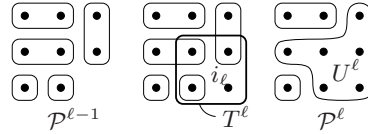

Figure 6: Computation of $\mathcal{P}^\ell$

Let $\boldsymbol{x}$ be a vector in $\mathrm{P}(f_\lambda)$. We say that a subset $S \subseteq V$ is $\boldsymbol{x}$-tight (with respect to $f_\lambda$) if $f_\lambda(S) = x(S)$. By the intersecting submodularity of $f_\lambda$, if $S$ and $T$ are intersecting and both $S$ and $T$ are $\boldsymbol{x}$-tight, then $S \cup T$ is also $\boldsymbol{x}$-tight. Using this property, we obtain the following property.

**Lemma 9.** *For each $\ell = 1, \ldots, n$, we have $\widehat{f}_\lambda(V^\ell) = x^\ell(V^\ell) = f_\lambda[\mathcal{P}^\ell]$.*

*Proof.* (Sketch) For each $\ell = 1, \ldots, n$, observe that $T^\ell$ is $\boldsymbol{x}^\ell$-tight. Thus, we can show by induction that any cluster in $\mathcal{P}^\ell$ is $\boldsymbol{x}^\ell$-tight for each $\ell = 1, \ldots, n$. Thus, $f_\lambda[\mathcal{P}^\ell] = \sum_{S \in \mathcal{P}^\ell} f_\lambda(S) = \sum_{S \in \mathcal{P}^\ell} x^\ell(S) = x^\ell(V^\ell)$. Moreover, the equality $\widehat{f}_\lambda(V^\ell) = x^\ell(V^\ell)$ follows from Theorem 8.   □

The procedure FINDPARTITION($\lambda$) returns $h_\lambda \in \mathbb{R}$ and $\mathcal{P}_\lambda$. By Theorem 8, we have $h_\lambda = h(\lambda)$, and by Lemma 9, we have $\widehat{f}_\lambda(V) = f_\lambda[\mathcal{P}_\lambda]$, and thus the partition $\mathcal{P}_\lambda$ of $V$ determines $h(\lambda)$. Clearly, the procedure runs in $\mathrm{O}(n \cdot \mathsf{SFM}(n))$ time. So, in the end, we completed the proof of Lemma 4.

## 5   Experimental results

### 5.1   Illustrative example

We first illustrate the proposed algorithm using two artificial datasets depicted in Figure 7. The above dataset is generated from four Gaussians with unit variance (whose centers are located at (3,3), (3,-3), (-3,3) and (-3,-3), respectively), and the below one consists of three cycles with different radii with a line. The numbers of samples in these examples are 100 and 310, respectively. Figure 7 shows the typical examples of partitions calculated through Algorithm SPLIT given in Section 3. Now the function $f$ is a cut function of a complete graph and the weight of each edge of that graph is determined by the Gaussian similarity function [15]. The values of $\lambda$ above the figures are the

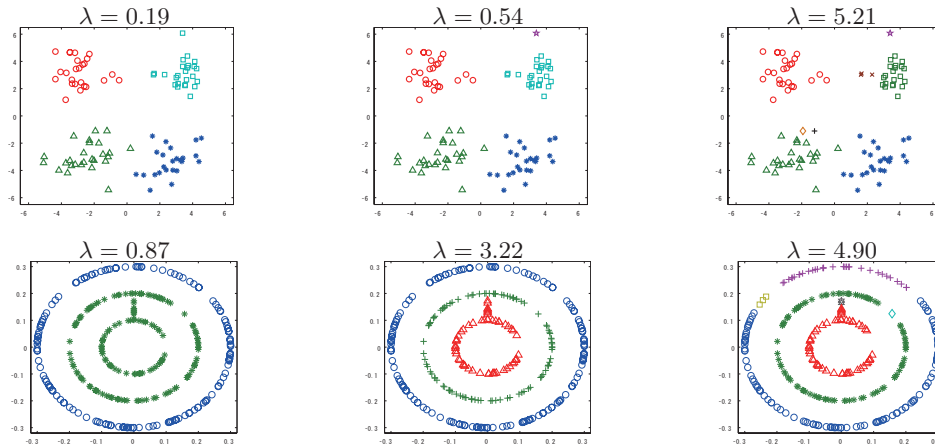

Figure 7: Illustrative examples with datasets from four Gaussians (above) and three circles (below).

ones identified as breakpoints. Note several partitions other than shown in the figures were obtained through one execution of Algorithm SPLIT. As can be seen, the algorithm produced several different sizes of clusters with inclusive relations.

## 5.2 Empirical comparison

Next, in this subsection, we empirically compare the performance of the algorithm with the existing algorithms using several synthetic and real world datasets from the UCI repository. The compared algorithms are k-means method, spectral-clustering method with normalized-cut [11] and maximum-margin clustering [16], and we used cut functions as the objective functions for the MAC clustering algorithm. The three UCI datasets used in this experiment are 'Glass', 'Iris' and 'Libras' which respectively consist of 214, 150 and 360 samples, respectively. For the existing algorithms, the number of clusters was selected through 5-fold cross-validation (again note that our algorithm needs no such hyper-parameter tuning). Table 1 shows the clustering accuracy when applying the algorithms to two artificial (stated in Subsection 5.1 and three UCI datasets. For our algorithm, the results with the best performance between among several partitions are shown. As can be seen, our algorithm seems to be competitive with the existing leading algorithms for these datasets.

|  | Gaussian | Circle | Iris | Libras | Glass |
|---|---|---|---|---|---|
| k-means | **1.0** | 0.88 | 0.79 | 0.85 | 0.93 |
| normalized cut | 0.88 | 0.86 | 0.84 | 0.87 | 0.93 |
| maximum margin | 0.99 | **1.0** | 0.96 | 0.90 | **0.97** |
| minimum average | 0.99 | **1.0** | **0.99** | **0.97** | **0.97** |

Table 1: Clustering accuracy for the proposed and existing algorithms.

## 6 Concluding remarks

We have introduced the new concept, the minimum average cost clustering problem. We have shown that the set of minimum average cost clusterings has a compact representation, and if the clustering cost is given by a submodular function, we have proposed a polynomial time algorithm that compute all information about minimum average cost clusterings. This result contrasts sharply with the NP-hardness of the optimal $k$-clustering problem [5]. The present paper reinforced the importance of the theory of intersecting submodular functions from the viewpoint of clustering.

### Acknowledgments

This work is supported in part by JSPS Global COE program "Computationism as a Foundation for the Sciences", KAKENHI (20310088, 22700007, and 22700147), and JST PRESTO program. We would also like to thank Takuro Fukunaga for his helpful comments.

## Footnotes

[1]To emphasize the difference between submodular and intersecting submodular functions, in what follows we refer to a submodular function as a *fully* submodular function.

# References

[1] W. H. Cunningham: Optimal attack and reinforcement of a network. *Journal of the ACM* **32** (1985), pp. 549–561.

[2] R. P. Dilworth: Dependence relations in a semimodular lattice. *Duke Mathematical Journal*, **11** (1944), pp. 575–587.

[3] J. Edmonds: Submodular functions, matroids, and certain polyhedra. *Combinatorial Structures and Their Applications*, R. Guy, H. Hanani, N. Sauer, and J. Schönheim, eds., Gordon and Breach, 1970, pp. 69–87.

[4] S. Fujishige: *Submodular Functions and Optimization* (Second Edition). Elsevier, Amsterdam, 2005.

[5] O. Goldschmidt and D. S. Hochbaum: A polynomial algorithm for the $k$-cut problem for fixed $k$, *Mathematics of Operations Research*, **19** (1994), pp. 24–37.

[6] S. Iwata: Submodular function minimization. *Mathematical Programming*, **112** (2008), pp. 45–64.

[7] V. Kolmogorov: A faster algorithm for computing the principal sequence of partitions of a graph. *Algorithmica* **56**, pp. 394-412.

[8] Y. Kawahara, K. Nagano, and Y. Okamoto: Submodular fractional programming for balanced clustering. Pattern Recognition Letters, to appear.

[9] M. Narasimhan and J. Bilmes: Local search for balanced submodular clusterings. In *Proceedings of the 12th International Joint Conference on Artificial Intelligence* (IJCAI 2007), pp. 981–986.

[10] M. Narasimhan, N. Jojic, and J. Bilmes: Q-clustering. In *Advances in Neural Information Processing Systems*, **18** (2006), pp. 979–986. Cambridge, MA: MIT Press.

[11] A. Y. Ng, M. I. Jordan, and Y. Weiss. On spectral clustering: Analysis and an algorithm. *Advances in neural information processing systems*, 2:849–856, 2002.

[12] K. Okumoto, T. Fukunaga, and H. Nagamochi: Divide-and-conquer algorithms for partitioning hypergraphs and submodular systems. In *Proceedings of the 20th International Symposium on Algorithms and Computation* (ISAAC 2009), LNCS 5878, 2009, pp. 55–64.

[13] M. Queyranne: Minimizing symmetric submodular functions, *Mathematical Programming*, **82** (1998), pp. 3–12.

[14] A. Schrijver: *Combinatorial Optimization — Polyhedra and Efficiency*. Springer-Verlag, 2003.

[15] U. von Luxburg: Tutorial on spectral clustering. *Statistics and Computing* **17** (2007), pp. 395–416.

[16] L. Xu, J. Neufeld, B. Larson, and D. Schuurmans. Maximum margin clustering. *Advances in neural information processing systems*, 17:1537–1544, 2005.

[17] L. Zhao, H. Nagamochi, and T. Ibaraki: Approximating the minimum $k$-way cut in a graph via minimum 3-way cuts. *Journal of Combinatorial Optimization*, **5** (2001), pp. 397–410.

[18] L. Zhao, H. Nagamochi, and T. Ibaraki: A unified framework for approximating multiway partition problems. In *Proceedings of the 12th International Symposium on Algorithms and Computation* (ISAAC 2001), LNCS 2223, 2001, pp. 682–694.

